# Learning to estimate scenes from images

**William T. Freeman and Egon C. Pasztor**
MERL, Mitsubishi Electric Research Laboratory
201 Broadway; Cambridge, MA 02139
freeman@merl.com, pasztor@merl.com

## Abstract

We seek the scene interpretation that best explains image data. For example, we may want to infer the projected velocities (scene) which best explain two consecutive image frames (image). From synthetic data, we model the relationship between image and scene patches, and between a scene patch and neighboring scene patches. Given a new image, we propagate likelihoods in a Markov network (ignoring the effect of loops) to infer the underlying scene. This yields an efficient method to form low-level scene interpretations. We demonstrate the technique for motion analysis and estimating high resolution images from low-resolution ones.

## 1 Introduction

There has been recent interest in studying the statistical properties of the visual world. Olshausen and Field [23] and Bell and Sejnowski [2] have derived V1-like receptive fields from ensembles of images; Simoncelli and Schwartz [30] account for contrast normalization effects by redundancy reduction. Li and Atick [1] explain retinal color coding by information processing arguments. Various research groups have developed realistic texture synthesis methods by studying the response statistics of V1-like multi-scale, oriented receptive fields [12, 7, 33, 29]. These methods help us understand the early stages of image representation and processing in the brain.

Unfortunately, they don't address how a visual system might *interpret* images, i.e., estimate the underlying scene. In this work, we study the statistical properties of a *labelled* visual world, images together with scenes, in order to infer scenes from images. The image data might be single or multiple frames; the scene quantities

to be estimated could be projected object velocities, surface shapes, reflectance patterns, or colors.

We ask: can a visual system correctly interpret a visual scene if it models (1) the probability that any local scene patch generated the local image, and (2) the probability that any local scene is the neighbor to any other? The first probabilities allow making scene estimates from local image data, and the second allow these local estimates to propagate. This leads to a Bayesian method for low level vision problems, constrained by Markov assumptions. We describe this method, and show it working for two low-level vision problems.

## 2   Markov networks for scene estimation

First, we synthetically generate images and their underlying scene representations, using computer graphics. The synthetic world should typify the visual world in which the algorithm will operate.

For example, for the motion estimation problem of Sect. 3, our training images were irregularly shaped blobs, which could occlude each other, moving in randomized directions at speeds up to 2 pixels per frame. The contrast values of the blobs and the background were randomized. The image data were the concatenated image intensities from two successive frames of an image sequence. The scene data were the velocities of the visible objects at each pixel in the two frames.

Second, we place the image and scene data in a Markov network [24]. We break the images and scenes into localized patches where image patches connect with underlying scene patches; scene patches also connect with neighboring scene patches. The neighbor relationship can be with regard to position, scale, orientation, etc.

For the motion problem, we represented both the images and the velocities in 4-level Gaussian pyramids [6], to efficiently communicate across space. Each scene patch then additionally connects with the patches at neighboring resolution levels. Figure 2 shows the multiresolution representation (at one time frame) for images and scenes.[1]

Third, we propagate probabilities. Weiss showed the advantage of belief propagation over regularization methods for several 1-d problems [31]; we apply related methods to our 2-d problems. Let the $i$th and $j$th image and scene patches be $y_i$ and $x_j$, respectively. For the MAP estimate [3] of the scene data,[2] we want to find $\text{argmax}_{x_1,x_2,...,x_N} P(x_1,x_2,...,x_N|y_1,y_2,...,y_M)$, where $N$ and $M$ are the number of scene and image patches. Because the joint probability is simpler to compute, we find, equivalently, $\text{argmax}_{x_1,x_2,...,x_N} P(x_1,x_2,...,x_N,y_1,y_2,...,y_M)$.

The conditional independence assumptions of the Markov network let us factorize the desired joint probability into quantities involving only local measurements and calculations [24, 32]. Consider the two-patch system of Fig. 1. We can factorize $P(x_1,x_2,y_1,y_2)$ in three steps: (1) $P(x_1,x_2,y_1,y_2) = P(x_2,y_1,y_2|x_1)P(x_1)$ (by elementary probability); (2) $P(x_2,y_1,y_2|x_1) = P(y_1|x_1)P(x_2,y_2|x_1)$ (by conditional

independence); (3) $P(x_2, y_2|x_1) = P(x_2|x_1)P(y_2|x_2)$ (by elementary probability and the Markov assumption). To estimate just $x_1$ at node 1, the $\text{argmax}_{x_2}$ becomes $\max_{x_2}$, and then slides over constants, giving terms involving only local computations at each node:

$$\text{argmax}_{x_1}\max_{x_2}P(x_1, x_2, y_1, y_2) = \text{argmax}_{x_1}[P(x_1)P(y_1|x_1)\max_{x_2}[P(x_2|x_1)P(y_2|x_2)]]. \tag{1}$$

This factorization generalizes to any network structure without loops. We use a *different* factorization at each scene node: we turn the initial joint probability into a conditional by factoring out *that* node's prior, $P(x_j)$, then proceeding analogously to the example above. The resulting factorized computations give local propagation rules, similar to those of [24, 32]: Each node, $j$, receives a message from each neighbor, $k$, which is an accumulated likelihood function, $L_{kj} = P(y_k \ldots y_z|x_j)$, where $y_k \ldots y_z$ are all image nodes that lie at or beyond scene node $k$, relative to scene node $j$. At each iteration, more image nodes $y$ enter that likelihood function. After each iteration, the MAP estimate at node $j$ is $\text{argmax}_{x_j}P(x_j)P(y_j|x_j)\prod_k L_{kj}$, where $k$ runs over all scene node neighbors of node $j$. We calculate $L_{kj}$ from:

$$L_{kj} = \max_{x_k}P(x_k|x_j)P(y_k|x_k)\prod_{l \neq j}\tilde{L}_{lk}, \tag{2}$$

where $\tilde{L}_{lk}$ is $L_{lk}$ from the previous iteration. The initial $\tilde{L}_{lk}$'s are 1. Using the

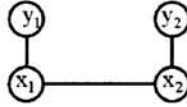

**Figure 1:** Markov network nodes used in example.

factorization rules described above, one can verify that the local computations will compute $\text{argmax}_{x_1, x_2, \ldots, x_N}P(x_1, x_2, \ldots, x_N|y_1, y_2, \ldots, y_M)$, as desired. To learn the network parameters, we measure $P(x_j)$, $P(y_j|x_j)$, and $P(x_k|x_j)$, directly from the synthetic training data.

If the network contains loops, the above factorization does not hold. Both learning and inference then require more computationally intensive methods [15]. Alternatively, one can use multi-resolution quad-tree networks [20], for which the factorization rules apply, to propagate information spatially. However, this gives results with artifacts along quad-tree boundaries, statistical boundaries in the model not present in the real problem. We found good results by including the loop-causing connections between adjacent nodes at the same tree level but applying the factorized propagation rules, anyway. Others have obtained good results using the same approach for inference [8, 21, 32]; Weiss provides theoretical arguments why this works for certain cases [32].

## 3 Discrete Probability Representation (motion example)

We applied the training method and propagation rules to motion estimation, using a vector code representation [11] for both images and scenes. We wrote a tree-structured vector quantizer, to code 4 by 4 pixel by 2 frame blocks of image data

for each pyramid level into one of 300 codes for each level. We also coded scene patches into one of 300 codes.

During training, we presented approximately 200,000 examples of irregularly shaped moving blobs, some overlapping, of a contrast with the background randomized to one of 4 values. Using co-occurance histograms, we measured the statistical relationships that embody our algorithm: $P(x)$, $P(y|x)$, and $P(x_n|x)$, for scene $x_n$ neighboring scene $x$.

Figure 2 shows an input test image, (a) before and (b) after vector quantization. The true underlying scene, the desired output, is shown (c) before and (d) after vector quantization. Figure 3 shows six iterations of the algorithm (Eq. 2) as it converges to a good estimate for the underlying scene velocities. The local probabilities we learned ($P(x)$, $P(y|x)$, and $P(x_n|x)$) lead to figure/ground segmentation, aperture problem constraint propagation, and filling-in (see caption).

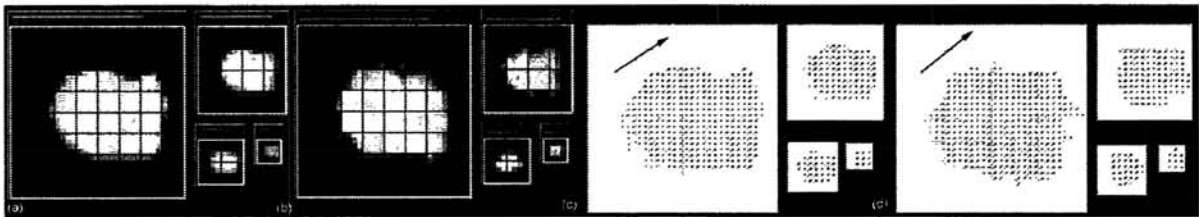

**Figure 2:** (a) First of two frames of image data (in gaussian pyramid), and (b) vector quantized. (c) The optical flow scene information, and (d) vector quantized. Large arrow added to show small vectors' orientation.

## 4   Density Representation (super-resolution example)

For super-resolution, the input "image" is the high-frequency components (sharpest details) of a sub-sampled image. The "scene" to be estimated is the high-frequency components of the full-resolution image, Fig. 4.

We improved our method for this second problem. A faithful image representation requires so many vector codes that it becomes infeasible to measure the prior and co-occurance statistics (note unfaithful fit of Fig. 2). On the other hand, a discrete representation allows fast propagation. We developed a hybrid method that allows both good fitting and fast propagation.

We describe the image and scene patches as vectors in a continuous space, and first modelled the probability densities, $P(x)$, $P(y,x)$, and $P(x_n,x)$, as gaussian mixtures [4]. (We reduced the dimensionality some by principal components analysis [4]). We then evaluated the prior and conditional distributions of Eq. 2 only at a discrete set of scene values, different for each node. (This sample-based approach relates to [14, 7]). The scenes were a sampling of those scenes which render to the image at that node. This focusses the computation to the locally feasible scene interpretations. $P(x_k|x_j)$ in Eq. 2 becomes the ratios of the gaussian mixtures $P(x_k,x_j)$ and $P(x_j)$, evaluated at the scene samples at nodes $k$ and $j$, respectively. $P(y_k|x_k)$ is $P(y_k,x_k)/P(x_k)$ evaluated at the scene samples of node $k$.

To select the scene samples, we could condition the mixture $P(y,x)$ on the $y$ observed at each node, and sample $x$'s from the resulting mixture of gaussians. We obtained somewhat better results by using the scenes from the training set whose

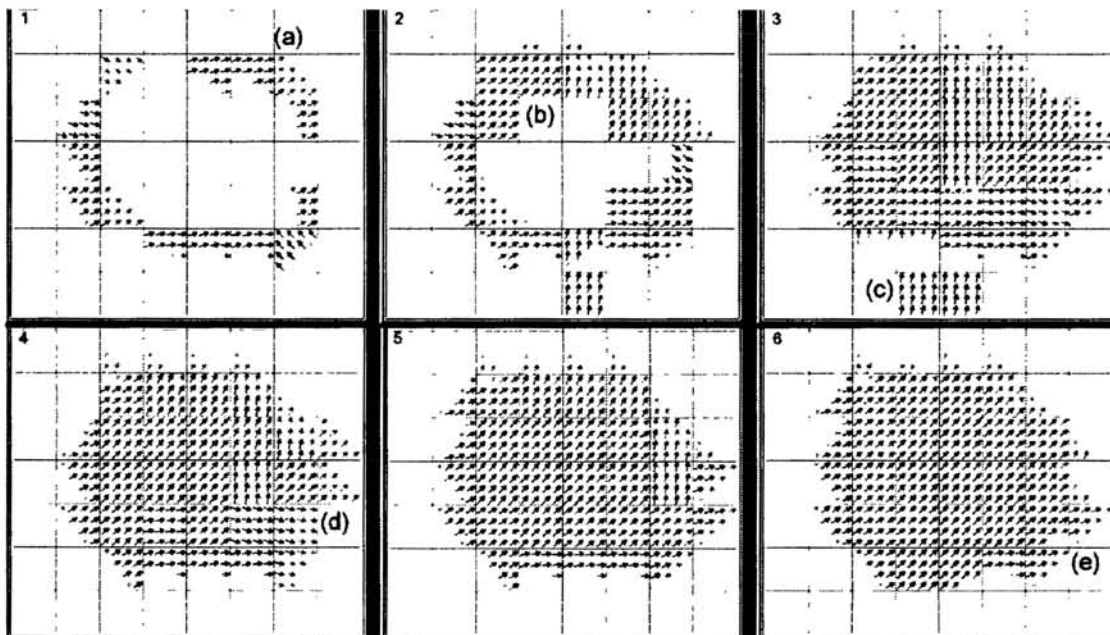

**Figure 3:** The most probable scene code for Fig. 2b at first 6 iterations of Bayesian belief propagation. (a) Note initial motion estimates occur only at edges. Due to the "aperture problem", initial estimates do not agree. (b) Filling-in of motion estimate occurs. Cues for figure/ground determination may include edge curvature, and information from lower resolution levels. Both are included implicitly in the learned probabilities. (c) Figure/ground still undetermined in this region of low edge curvature. (d) Velocities have filled-in, but do not yet all agree. (e) Velocities have filled-in, and agree with each other and with the correct velocity direction, shown in Fig. 2.

images most closely matched the image observed at that node (thus avoiding one gaussian mixture modeling step).

Using 40 scene samples per node, setting up the $P(x_k|x_j)$ matrix for each link took several minutes for 96x96 pixel images. The scene (high resolution) patch size was 3x3; the image (low resolution) patch size was 7x7. We didn't feel long-range scene propagation was critical here, so we used a flat, not a pyramid, node structure. Once the matrices were computed, the iterations of Eq. 2 were completed within seconds.

Figure 4 shows the results. The training images were random`shaded and painted blobs such as the test image shown. After 5 iterations, the synthesized maximum likelihood estimate of the high resolution image is visually close to the actual high frequency image (top row). (Including $P(x)$ gave too flat results, we suspect due to errors modeling that highly peaked distribution). The dominant structures are all in approximately the correct position. This may enable high quality zooming of low-resolution images, attempted with limited success by others [28, 25].

## 5   Discussion

In related applications of Markov random fields to vision, researchers typically use relatively simple, heuristically derived expressions (rather than learned) for the likelihood function $P(y|x)$ or for the spatial relationships in the prior term on scenes

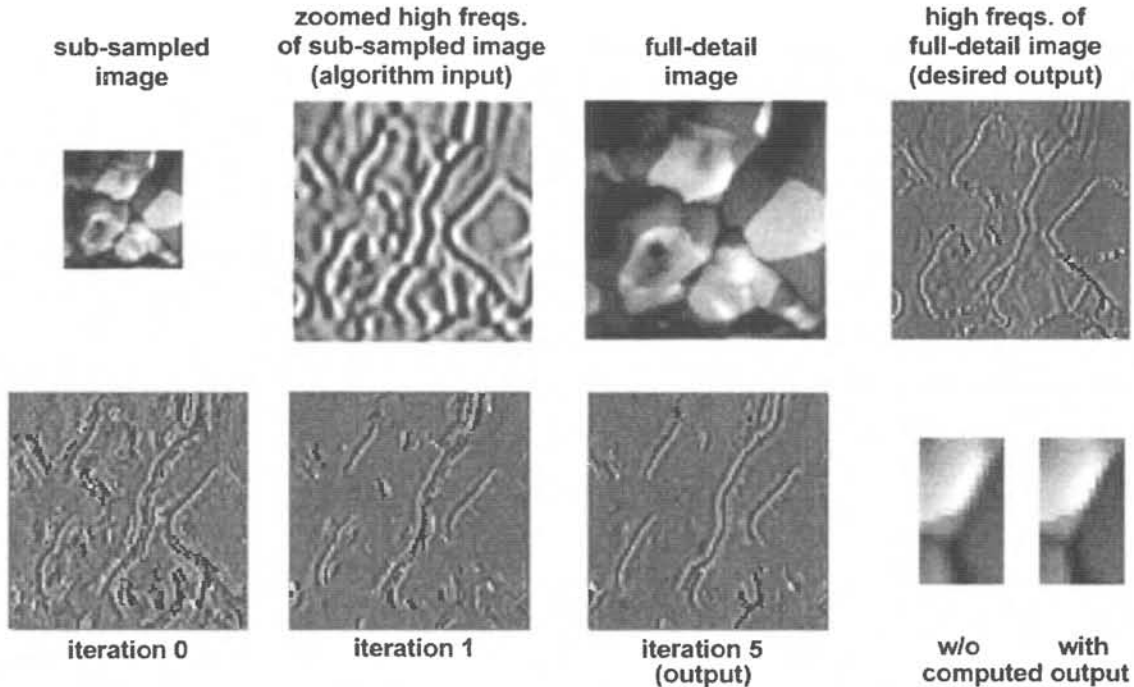

**Figure 4:** Superresolution example. Top row: Input and desired output (contrast normalized, only those orientations around vertical). Bottom row: algorithm output and comparison of image with and without estimated high vertical frequencies.

[10, 26, 9, 17, 5, 20, 19, 27]. Some researchers have applied related learning approaches to low-level vision problems, but restricted themselves to linear models [18, 13]. For other learning or constraint propagation approaches in motion analysis, see [20, 22, 16].

In summary, we have developed a principled and practical learning based method for low-level vision problems. Markov assumptions lead to factorizing the posterior probability. The parameters of our Markov random field are probabilities specified by the training data. For our two examples (programmed in C and Matlab, respectively), the training can take several hours but the running takes only several minutes. Scene estimation by Markov networks may be useful for other low-level vision problems, such as extracting intrinsic images from line drawings or photographs.

**Acknowledgements** We thank E. Adelson, J. Tenenbaum, P. Viola, and Y. Weiss for helpful discussions.

## Footnotes

[1]To maintain the desired conditional independence relationships, we appended the image data to the scenes. This provided the scene elements with image contrast information, which they would otherwise lack.

[2]Related arguments follow for the MMSE or other estimators.

## References

[1] J. J. Atick, Z. Li, and A. N. Redlich. Understanding retinal color coding from first principles. *Neural Computation*, 4:559–572, 1992.

[2] A. J. Bell and T. J. Senjowski. The independent components of natural scenes are edge filters. *Vision Research*, 37(23):3327–3338, 1997.

[3] J. O. Berger. *Statistical decision theory and Bayesian analysis.* Springer, 1985.

[4] C. M. Bishop. *Neural networks for pattern recognition.* Oxford, 1995.

[5] M. J. Black and P. Anandan. A framework for the robust estimation of optical flow. In *Proc. 4th Intl. Conf. Computer Vision*, pages 231–236. IEEE, 1993.

[6] P. J. Burt and E. H. Adelson. The Laplacian pyramid as a compact image code. *IEEE Trans. Comm.*, 31(4):532–540, 1983.

[7] J. S. DeBonet and P. Viola. Texture recognition using a non-parametric multi-scale

statistical model. In *Proc. IEEE Computer Vision and Pattern Recognition*, 1998.

[8] B. J. Frey. *Bayesian networks for pattern classification*. MIT Press, 1997.

[9] D. Geiger and F. Girosi. Parallel and deterministic algorithms from MRF's: surface reconstruction. *IEEE Pattern Analysis and Machine Intelligence*, 13(5):401–412, May 1991.

[10] S. Geman and D. Geman. Stochastic relaxation, Gibbs distribution, and the Bayesian restoration of images. *IEEE Pattern Analysis and Machine Intelligence*, 6:721–741, 1984.

[11] R. M. Gray, P. C. Cosman, and K. L. Oehler. Incorporating visual factors into vector quantizers for image compression. In A. B. Watson, editor, *Digital images and human vision*. MIT Press, 1993.

[12] D. J. Heeger and J. R. Bergen. Pyramid-based texture analysis/synthesis. In *ACM SIGGRAPH*, pages 229–236, 1995. In *Computer Graphics* Proceedings, Annual Conference Series.

[13] A. C. Hurlbert and T. A. Poggio. Synthesizing a color algorithm from examples. *Science*, 239:482–485, 1988.

[14] M. Isard and A. Blake. Contour tracking by stochastic propagation of conditional density. In *Proc. European Conf. on Computer Vision*, pages 343–356, 1996.

[15] M. I. Jordan, editor. *Learning in graphical models*. MIT Press, 1998.

[16] S. Ju, M. J. Black, and A. D. Jepson. Skin and bones: Multi-layer, locally affine, optical flow and regularization with transparency. In *Proc. IEEE Computer Vision and Pattern Recognition*, pages 307–314, 1996.

[17] D. Kersten. Transparancy and the cooperative computation of scene attributes. In M. S. Landy and J. A. Movshon, editors, *Computational Models of Visual Processing*, chapter 15. MIT Press, Cambridge, MA, 1991.

[18] D. Kersten, A. J. O'Toole, M. E. Sereno, D. C. Knill, and J. A. Anderson. Associative learning of scene parameters from images. *Applied Optics*, 26(23):4999–5006, 1987.

[19] D. Knill and W. Richards, editors. *Perception as Bayesian inference*. Cambridge Univ. Press, 1996.

[20] M. R. Luettgen, W. C. Karl, and A. S. Willsky. Efficient multiscale regularization with applications to the computation of optical flow. *IEEE Trans. Image Processing*, 3(1):41–64, 1994.

[21] D. J. C. Mackay and R. M. Neal. Good error–correcting codes based on very sparse matrices. In *Cryptography and coding – LNCS 1025*, 1995.

[22] S. Nowlan and T. J. Senjowski. A selection model for motion processing in area MT of primates. *J. Neuroscience*, 15:1195–1214, 1995.

[23] B. A. Olshausen and D. J. Field. Emergence of simple-cell receptive field properties by learning a sparse code for natural images. *Nature*, 381:607–609, 1996.

[24] J. Pearl. *Probabilistic reasoning in intelligent systems: networks of plausible inference*. Morgan Kaufmann, 1988.

[25] A. Pentland and B. Horowitz. A practical approach to fractal-based image compression. In A. B. Watson, editor, *Digital images and human vision*. MIT Press, 1993.

[26] T. Poggio, V. Torre, and C. Koch. Computational vision and regularization theory. *Nature*, 317(26):314–139, 1985.

[27] E. Saund. Perceptual organization of occluding contours of opaque surfaces. In *CVPR '98 Workshop on Perceptual Organization*, Santa Barbara, CA, 1998.

[28] R. R. Schultz and R. L. Stevenson. A Bayesian approach to image expansion for improved definition. *IEEE Trans. Image Processing*, 3(3):233–242, 1994.

[29] E. P. Simoncelli. Statistical models for images: Compression, restoration and synthesis. In *31st Asilomar Conf. on Sig., Sys. and Computers*, Pacific Grove, CA, 1997.

[30] E. P. Simoncelli and O. Schwartz. Modeling surround suppression in v1 neurons with a statistically-derived normalization model. In *Adv. in Neural Information Processing Systems*, volume 11, 1999.

[31] Y. Weiss. Interpreting images by propagating Bayesian beliefs. In *Adv. in Neural Information Processing Systems*, volume 9, pages 908–915, 1997.

[32] Y. Weiss. Belief propagation and revision in networks with loops. Technical Report 1616, AI Lab Memo, MIT, Cambridge, MA 02139, 1998.

[33] S. C. Zhu and D. Mumford. Prior learning and Gibbs reaction-diffusion. *IEEE Pattern Analysis and Machine Intelligence*, 19(11), 1997.